# 'N-Body' Problems in Statistical Learning

**Alexander G. Gray**
Department of Computer Science
Carnegie Mellon University
*agray@cs.cmu.edu*

**Andrew W. Moore**
Robotics Inst. and Dept. Comp. Sci.
Carnegie Mellon University
*awm@cs.cmu.edu*

## Abstract

We present efficient algorithms for all-point-pairs problems, or 'N-body'-like problems, which are ubiquitous in statistical learning. We focus on six examples, including nearest-neighbor classification, kernel density estimation, outlier detection, and the two-point correlation. These include any problem which abstractly requires a comparison of each of the $N$ points in a dataset with each other point and would naively be solved using $N^2$ distance computations. In practice $N$ is often large enough to make this infeasible. We present a suite of new geometric techniques which are applicable in principle to any 'N-body' computation including large-scale mixtures of Gaussians, RBF neural networks, and HMM's. Our algorithms exhibit favorable asymptotic scaling and are empirically several orders of magnitude faster than the naive computation, even for small datasets. We are aware of no exact algorithms for these problems which are more efficient either empirically or theoretically. In addition, our framework yields simple and elegant algorithms. It also permits two important generalizations beyond the standard all-point-pairs problems, which are more difficult. These are represented by our final examples, the multiple two-point correlation and the notorious n-point correlation.

## 1 Introduction

This paper is about accelerating a wide class of statistical methods that are naively quadratic in the number of datapoints. [1] We introduce a family of dual *kd*-tree traversal algorithms for these problems. They are the statistical siblings of powerful state-of-the-art $N$-body simulation algorithms [1, 4] of computational physics, but the computations within statistical learning present new opportunities for acceleration and require techniques more general than those which have been exploited for the special case of potential-based problems involving forces or charges.

We describe in detail a dual-tree algorithm for calculating the two-point correlation, the simplest case of the problems we consider; for the five other statistical problems we consider, we show only performance results for lack of space. The last of our examples,

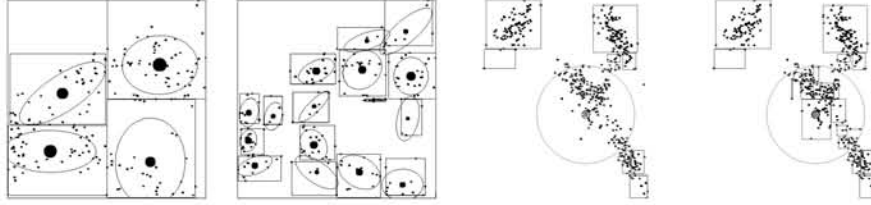

Figure 1: A *kd*-tree. (a) Nodes at level 3. (b) Nodes at level 5. The dots are the individual data points. The sizes and positions of the disks show the node counts and centroids. The ellipses and rectangles show the covariances and bounding boxes. (c) The rectangles show the nodes pruned during a RangeSearch for one (depicted) query and radius. (d) More pruning is possible using RangeCount instead of RangeSearch.

the $n$-point correlation, illustrates a generalization from all-point-pairs problems to all-$n$-tuples problems, which are much harder (naively $O(N^n)$). For all the examples, we believe there exist no exact algorithms which are faster either empirically or theoretically, nor any approximate algorithms that are faster while providing guarantees of acceptably high accuracy (as ours do). For $n$-tuple $N$-body problems in particular, this type of algorithm design appears to have surpassed the existing computational barriers. In addition, all the algorithms in this paper can be compactly defined and are easy to implement.

**Statistics and geometry.** We proceed by viewing these statistical problems as geometric problems, exploiting the data's hyperstructure. Each algorithm utilizes Multiresolution *kd*-trees, providing a geometric partitioning of the data space which is used to reason about entire chunks of the data simultaneously.

**A review of *kd*-trees and *mrkd*-trees.** A *kd*-tree [3] records a $d$-dimensional data set containing $N$ records. Each node represents a set of data points by their bounding box. Non-leaf nodes have two children, obtained by splitting the widest dimension of the parent's bounding box. For the purposes of this paper, nodes are split until they contain only one point, where they become leaves. An *mrkd*-tree [2, 6] is a conventional *kd*-tree decorated, at each node, with extra statistics about the node's data, such as their count, centroid, and covariance. They are an instance of the idea of cached sufficient statistics [8] and are quite efficient in practice. [2] See Figure 1.

## 2 The 2-point correlation function

The two-point correlation is a spatial statistic which is of fundamental importance in many natural sciences, in particular astrophysics and biology. It can be thought of roughly as a measure of the clumpiness of a set of points. It is easily defined as the number of pairs of points in a dataset which lie within a given radius $r$ of each other.

### 2.1 Previous approaches

**Quadratic algorithm.** The most naive approach is to simply compare each datum to each other one, incrementing a count if the distance between them is less than $r$. This has $O(N^2)$ cost, unacceptably high for problems of practical interest.

**Binning and gridding algorithms.** The schemes in widespread use [12, 13] are mainly of this sort. The idea of binning is simply to divide the data space into a fine grid defining a set of bins, perform the quadratic algorithm on the bins as if they were individual data, then multiply by the bin sizes as appropriate to get an estimate of the total count. The idea of gridding is to divide the data space into a coarse grid, perform the quadratic algorithm within each bin, and sum the results over all bins to get an estimate of the total count. These are both of course very approximate methods yielding large errors. They are not usable when $r$ is small or $r$ is large, respectively.

**Range-searching with a *kd*-tree.** An approach to the two-point correlation computation that has been taken is to treat it as a *range-searching* problem [5, 10], since *kd*-trees have been historically almost synonymous with range-searching. The idea is that we will make each datapoint in turn a *query point* and then execute a range search of the *kd*-tree to find all other points within distance $r$ of the query. A search is a depth-first traversal of the *kd*-tree, always checking the minimum possible distance $d_{min}$ between the query and the hyper-rectangle surrounding the current node. If $d_{min} > r$ there is no point in visiting the node's children, and computation is saved. We call this *exclusion-based* pruning.

The range searching avoids computing most of the distances between pairs of points further than $r$ apart, which is a considerable saving if $r$ is small. But is it the best we can do? And what if $r$ is large? We now propose several layers of new approaches.

## 2.2 Better geometric approaches: new algorithms

**Single-tree search** (Range-Counting Algorithm). A straightforward extension can exploit the fact that unlike conventional use of range searching, these statistics frequently don't need to retrieve all the points in the radius but merely to count them. The *mrkd*-tree has, in each node, the count of the number of data it contains—the simplest kind of cached sufficient statistic. At a given node, if the distance between the query and the farthest point of the bounding box of the data in the node is smaller than the radius $r$, clearly every datum in the node is within range of the query. We can then simply add the node's stored count to the total count. We call this *subsumption*. [3] (Note that both exclusion and subsumption are simple computations because the geometric regions are always axis-parallel rectangles.) This paper introduces new single-tree algorithms for most of our examples, though it is not our main focus.

**Dual-tree search.** This is the primary topic of this paper. The idea is to consider the query points in chunks as well, as defined by nodes in a *kd*-tree. In the general case where the query points are different from the data being queried, a separate *kd*-tree is built for the query points; otherwise a query node and a data node are simply pointers into the same *kd*-tree. Dual-tree search can be thought of as a simultaneous traversal of two trees, instead of iterating over the query points in an outer loop and only exploiting single-tree-search in the inner loop. Dual-tree search is based on node-node comparisons while Single-tree search was based on point-node comparisons.

Pseudocode for a recursive procedure called **TwoPoint**() is shown in Figure 2. It counts the number of pairs of points ($x_q \in$ QNODE, $x_d \in$ DNODE) such that $|x_q - x_d| < r$. Before doing any real work, the procedure checks whether it can perform an exclusion pruning (in which case the call terminates, returning 0) or subsumption pruning (in which case the call terminates, returning the product of the number of points in the two nodes). If neither of these prunes occur, then depending on whether QNODE and/or DNODE are leaves, the corresponding recursive calls are made.

```
TwoPoint(QNODE,DNODE,r)
  if excludes(QNODE,DNODE,r), return;

  if subsumes(QNODE,DNODE,r)
    total = total + ( count(QNODE) × count(DNODE) ); return;

  if leaf(QNODE) and leaf(DNODE)
    if distance(QNODE,DNODE) < r, total = total + 1;

  if leaf(QNODE) and notleaf(DNODE)
    TwoPoint(QNODE,leftchild(DNODE),r); TwoPoint(QNODE,rightchild(DNODE),r);

  if notleaf(QNODE) and leaf(DNODE)
    TwoPoint(leftchild(QNODE),DNODE,r); TwoPoint(rightchild(QNODE),DNODE,r);

  if notleaf(QNODE) and notleaf(DNODE)
    TwoPoint(leftchild(QNODE),leftchild(DNODE),r); TwoPoint(leftchild(QNODE),rightchild(DNODE),r);
    TwoPoint(rightchild(QNODE),leftchild(DNODE),r); TwoPoint(rightchild(QNODE),rightchild(DNODE),r);
```

Figure 2: A recursive Dual-tree code. All the reported algorithms have a similar brevity.

Importantly, both kinds of prunings can now apply to many query points at once, instead of each nearby query point rediscovering the same prune during the Single-tree search. The intuition behind Dual-tree's advantage can be seen by considering two cases. First, if $r$ is so large that all pairs of points are counted then the Single-Tree search will perform $O(N)$ operations, where each query point immediately prunes at the root, while Dual-Tree search will perform $O(1)$ operations. Second, if $r$ is so small that no pairs of points are counted, Single-Tree search will run to one leaf for each query, meaning total work $O(N \log N)$ whereas Dual-tree search will visit each leaf once, meaning $O(N)$ work. Note, however, that in the middle case of a medium-size $r$, Dual-tree is theoretically only a constant-factor superior to Single-tree. [4]

**Non-redundant dual-tree search.** So far, we have discussed two operations which cut short the need to traverse the tree further - exclusion and subsumption. Another form of pruning is to eliminate node-node comparisons which have been performed already in the reverse order. This can be done [11] simply by (virtually) ranking the datapoints according to their position in a depth-first traversal of the tree, then recording for each node the minimum and maximum ranks of the points it owns, and pruning whenever QNODE's maximum rank is less than DNODE's minimum rank. This is useful for all-pairs problems, but becomes *essential* for all-$n$-tuples problems. This kind of pruning is not practical for Single-tree search. Figure 3 shows the performance of a two-point correlation algorithm using all the aforementioned pruning methods.

**Multiple radii simultaneously.** Most often in practice, the two-point is computed for many successive radii so that a curve can be plotted, indicating the clumpiness on different scales. Though the method presented so far is fast, it may have to be run once for each of, say, 1,000 radii. It is possible to perform a single, faster computation for all the radii simultaneously, by taking advantage of the nesting structure of the ordered radii, with an algorithm which recursively narrows the radii which still need to

| Algorithm | # Data | Quadratic | Single-tree | Dual-tree | ST Speedup | DT Speedup |
|-----------|--------|-----------|-------------|-----------|------------|------------|
| twopoint | 10,000 | 132 | 2.2 | 1.2 | 60 | 110 |
| twopoint | 50,000 | 3300 est. | 11.8 | 7.0 | 280 | 471 |
| twopoint | 150,000 | 30899 est. | 37 | 20 | 835 | 1545 |
| twopoint | 300,000 | 123599 est. | 76 | 40 | 1626 | 3090 |
| nearest | 10,000 | 139 | 2.0 | 1.4 | 70 | 99 |
| nearest | 20,000 | 556 est. | 11.6 | 9.8 | 48 | 57 |
| nearest | 50,000 | 3475 est. | 30.6 | 26.4 | 114 | 132 |
| outliers | 10,000 | 141 | 2.3 | 1.2 | 61 | 118 |
| outliers | 50,000 | 3525 est. | 12 | 6.5 | 294 | 542 |
| outliers | 150,000 | 33006 est. | 36 | 21 | 917 | 1572 |
| outliers | 300,000 | 132026 est. | 72 | 44 | 1834 | 3001 |

Figure 3: Our experiments timed our algorithms on large astronomical datasets of current scientific interest, consisting of x-y positions of sky objects from the Sloane Digital Sky Survey. All times are given in seconds, and runs were performed on a Pentium III-500 MHz Linux workstation. The larger runtimes for the quadratic algorithm were estimated based on those for smaller datasets. The dual $k$d-tree method is about a factor of 2 faster than the single $k$d-tree method, and both are 3 orders of magnitude faster than the quadratic method for a medium-sized dataset of 300,000 points.

| # Data | 1 | 100 | 1000 | Speedup |
|--------|-----|------|------|---------|
| 10,000 | 1.2 | 1.8 | 2.4 | 500 |
| 20,000 | 2.8 | 6.4 | 6.6 | 424 |
| 50,000 | 7.0 | 31 | 31 | 226 |
| 150,000 | 20 | 133 | 146 | 137 |

| # Data | Quadratic | $10^{-2}$ | $10^{-6}$ | Speedup |
|--------|-----------|-----------|-----------|---------|
| 10,000 | 226 | 1.2 | 3.0 | 188 |
| 50,000 | 5650 est. | 10.4 | 16.8 | 543 |
| 150,000 | 50850 est. | 32 | 65 | 1589 |
| 300,000 | 203400 est. | 73 | 151 | 2786 |

Figure 4: (a) Runtimes for multiple 2-point correlation with increasing number of radii, and the speedup factored compared to 1,000 separate Dual-tree 2-point correlations. (b) Runtimes for kernel density estimation with decreasing levels of approximation, controlled by parameter $\epsilon$, and speedup over quadratic.

be considered based on the current closest and farthest distances between the nodes. The details are omitted for space, regrettably. The results in Figure 4 confirm that the algorithm quickly focuses on the radii of relevance: for 150,000 data, computing 1,000 2-point correlations took only 7 times as long as computing one.

## 3   Kernel density estimation

**Approximation accelerations.** A fourth major type of pruning opportunity is *approximation*. This is often needed in all-point-pairs computations which involve computing some real-valued function $f(x, y)$ between every pair of points $x$ and $y$. An example is kernel density estimation with an infinite-tailed kernel such as a Gaussian, in which every training point has some non-zero (though perhaps infinitesimal) contribution to the density at each test point.

For each query point $x_q$ we need to accumulate $K \sum_i w(|x_q - x_i|)$ where $K$ is a normalizing constant and $w$ is a weighting function (which we will need to assume is monotonic). A recursive call of the Dual-tree implementation has the following job: for $x_q \in$ QNODE compute the contribution to $x_q$'s summed weights that are due to all points in DNODE. Once again, before doing any real work we use simple rectangle geometry to compute the shortest and furthest possible distances between any $(x_q, x_d)$ pair. This bounds the minimum and maximum possible values of $Kw(|x_q - x_d|)$. If these bounds are tight enough (according to an approximation parameter $\epsilon$) we prune by simply distributing the midpoint weight to all the points in QNODE.

| $n$ | Naive | $n$-tree | Speedup |
|---|---|---|---|
| 2 | 528 | 2 | 264 |
| 3 | $1 \times 10^7$ est. | 186 | 56774 |
| 4 | $2 \times 10^{11}$ est. | 14441 | $1.4 \times 10^7$ |

| # Data | Time |
|---|---|
| 1000 | 1 |
| 2000 | 13 |
| 10000 | 1470 |
| 20000 | 14441 |

| $d$ | $n = 2$ | $n = 3$ | $n = 4$ |
|---|---|---|---|
| 1 | $< 1$ | $< 1$ | $< 1$ |
| 2 | $< 1$ | 3 | 23 |
| 3 | $< 1$ | 6 | 57 |
| 4 | $< 1$ | 7 | 73 |

Figure 5: (a) Runtimes for approximate $n$-point correlation with $\epsilon = 0.02$ and 20,000 data. (b) Runtimes for approximate 4-point with $\epsilon = 0.02$ and increasing data size. (c) Runtimes for exact $n$-point, run on 2000 datapoints of galaxies in $d$-dimensional color space.

# 4 The $n$-point correlation, for $n > 2$

The $n$-point correlation is the generalization of the 2-point correlation, which counts the number of $n$-tuples of points lying within radius $r$ of each other, or more generally, between some $r_{\min}$ and $r_{\max}$. [5] The implementation is entirely analogous to the 2-point case, using $n$ trees in general instead of two, except that there is more benefit in being careful about which of $2^n$ possible recursive calls to choose in the cases where you cannot prune, the approximation versions are harder, there is no immediately analogous Single-tree version of the algorithm, and anti-redundancy pruning is much more important. Figure 5 shows the unprecedented efficiency gains, which become more dramatic as $n$ increases.

**Approximating 'exact' computations.** Even for algorithms such as 2-point, that return exact counts, bounded approximation is possible. Suppose the true value of the 2-point function is $V^\star$ but that we can tolerate a fractional error of $\epsilon$: we'll accept any value $\hat{V}$ such that $|\hat{V} - V^\star| < \epsilon V^\star$. It is possible to adapt the dual-tree algorithm using a best-first iterative deepening search strategy to *guarantee* this result while exploiting permission to approximate effectively by building the count as much as possible from "easy-win" node pairs while doing approximation at hard deep node-pairs.

# 5 Outlier detection, nearest neighbors, and other problems

One of the main intents of this paper is to point out the broad applicability of this type of algorithm within statistical learning. Figure 3 shows performance results for our outlier detection and nearest neighbors algorithms. Figure 6 lists many $N$-body problems which are clear candidates for acceleration in future work. [6]

| Statistical Operation | Results here? | Approximation? | What is $N$? |
|---|---|---|---|
| 2-point function | Yes | Optional | # Data |
| $n$-point function | Yes | Optional | # Data |
| Multiple 2-point function | Yes | Optional | # Data |
| Batch $k$-nearest neighbor | Yes | Optional | # Data |
| Non-parameteric outlier detection/denoising | Yes | Optional | # Data |
| Batch Kernel density/classify/regression | Yes | Yes | # Data |
| Batch locally weighted regression | No | Yes | # Data |
| Batch kernel PCA | No | Yes | # Data |
| Gaussian process learning and prediction | No | Yes | # Data |
| $K$-means | No | Optional | # Data, Clusters |
| Mixture of Gaussians clustering | No | Yes | # Data, Clusters |
| Hidden Markov model | No | Yes | # Data, States |
| RBF neural network | No | Yes | # Data, Neurons |
| Finding pairs of correlated attributes | No | Optional | # Attributes |
| Finding $n$-tuples of correlated attributes | No | Optional | # Attributes |
| Dependency-tree learning | No | Optional | # Attributes |

Figure 6: A very brief sample of applicability of Dual-tree search methods.

## Footnotes

[1]In the general case, when we are computing distances between two different datasets having sizes $N_1$ and $N_2$, as in nearest-neighbor classification with separate training and test sets, say, the cost is $O(N_1 N_2)$.

[2] *mrkd*-trees can be built quickly, in time $O(dN \log N + d^2 N)$. Although we have not needed to do so, they can modified to become disk-resident for data sets with billions of records, and they can be efficiently updated incrementally. They scale poorly to higher dimensions but recent work [7] significantly remedies the dimensionality problem.

[3] Subsumption can also be exploited when other *aggregate* statistics, such as centroids or covariances of sets of points in a range are required [2, 14, 9].

[4]We'll summarize the asymptotic analysis briefly. If the data is uniformly distributed in $d$-dimensional space, the cost of computing the $n$-point correlation function on a dataset with $N$ points using the Dual-tree ($n$-tree) algorithm is $O(N^{\alpha_{nd}})$ where $\alpha_{nd}$ is the dimensionality of the manifold of $n$-tuples that are just on the border between being matched and not-matched, and is $\alpha_{nd} = n' \left(1 - \frac{n'-1}{2d}\right)$ where $n' = \min(n, d)$ For example, the 2-point correlation function in two dimensions is $O(N^{3/2})$, considerably better than the $O(N^2)$ naive algorithm. Disappointingly, for 2-point, this performance is asymptotically the same cost as Single-tree. For $n > 2$ our algorithm *is* better. Furthermore, if we can accept an approximate answer, the cost is $\left(\frac{n\alpha_{nd}}{\epsilon}\right)^{(\alpha_{nd}/(n-\alpha_{nd}))}$ which is independent of $N$.

[5] The $n$-point correlation is useful for detailed characterizations of mass distributions (including galaxies and biomasses). Higher-order $n$-point correlations detect increasingly subtle differences in mass distribution, and are also useful for assessing variance in the lower-order $n$-point statistics. For example, the three-point correlation, which measures the number of triplets of points meeting the specified geometric constraints, can distinguish between two distributions that have the same 2-point correlations but differ in their degree of "stripiness" versus "spottiness".

[6] In our nearest neighbors algorithm we consider the problem of finding, for each query point, its single nearest neighbor among the data points. (This is exactly the all-nearest-neighbors problem of computational geometry.) The methods are easily generalized to the case of finding the $k$ nearest neighbors, as in $k$-NN classification and locally weighted regression. Outlier detection is one of the most common statistical operations encountered in data analysis. The question of which procedure is most correct is an open and active one. We present here a natural operation which might be used directly for outlier detection, or within another procedure: for each of the points, find the number of other points that are within distance $r$ of it - those having zero neighbors within $r$ are defined as outliers. (This is exactly the all-range-count problem.)

# References

[1] J. Barnes and P. Hut. A Hierarchical $O(NlogN)$ Force-Calculation Algorithm. *Nature*, 324, 1986.

[2] K. Deng and A. W. Moore. Multiresolution instance-based learning. In *Proceedings of the Twelfth International Joint Conference on Artificial Intelligence*, pages 1233–1239, San Francisco, 1995. Morgan Kaufmann.

[3] J. H. Friedman, J. L. Bentley, and R. A. Finkel. An algorithm for finding best matches in logarithmic expected time. *ACM Transactions on Mathematical Software*, 3(3):209–226, September 1977.

[4] L. Greengard and V. Rokhlin. A Fast Algorithm for Particle Simulations. *Journal of Computational Physics*, 73, 1987.

[5] D. E. Knuth. *Sorting and Searching*. Addison Wesley, 1973.

[6] A. W. Moore. Very fast mixture-model-based clustering using multiresolution kd-trees. In M. Kearns and D. Cohn, editors, *Advances in Neural Information Processing Systems 10*, pages 543–549, San Francisco, April 1999. Morgan Kaufmann.

[7] A. W. Moore. The Anchors Hierarchy: Using the triangle inequality to survive high dimensional data. In *Twelfth Conference on Uncertainty in Artificial Intelligence (to appear)*. AAAI Press, 2000.

[8] A. W. Moore and M. S. Lee. Cached Sufficient Statistics for Efficient Machine Learning with Large Datasets. *Journal of Artificial Intelligence Research*, 8, March 1998.

[9] D. Pelleg and A. W. Moore. Accelerating Exact $k$-means Algorithms with Geometric Reasoning. In *Proceedings of the Fifth International Conference on Knowledge Discovery and Data Mining*. AAAI Press, 1999.

[10] F. P. Preparata and M. Shamos. *Computational Geometry*. Springer-Verlag, 1985.

[11] A. Szalay. Personal Communication. 2000.

[12] I. Szapudi. A New Method for Calculating Counts in Cells. *The Astrophysical Journal*, 1997.

[13] I. Szapudi, S. Colombi, and F. Bernardeau. Cosmic Statistics of Statistics. *Monthly Notices of the Royal Astronomical Society*, 1999.

[14] T. Zhang, R. Ramakrishnan, and M. Livny. BIRCH: An Efficient Data Clustering Method for Very Large Databases. In *Proceedings of the Fifteenth ACM SIGACT-SIGMOD-SIGART Symposium on Principles of Database Systems : PODS 1996*. Assn for Computing Machinery, 1996.
